# Factorial LDA:
# Sparse Multi-Dimensional Text Models

**Michael J. Paul and Mark Dredze**
Human Language Technology Center of Excellence (HLTCOE)
Center for Language and Speech Processing (CLSP)
Johns Hopkins University
Baltimore, MD 21218
{mpaul,mdredze}@cs.jhu.edu

## Abstract

Latent variable models can be enriched with a **multi-dimensional** structure to consider the many latent factors in a text corpus, such as topic, author perspective and sentiment. We introduce *factorial LDA*, a multi-dimensional model in which a document is influenced by $K$ different factors, and each word token depends on a $K$-dimensional vector of latent variables. Our model incorporates structured word priors and learns a sparse product of factors. Experiments on research abstracts show that our model can learn latent factors such as research topic, scientific discipline, and focus (methods vs. applications). Our modeling improvements reduce test perplexity and improve human interpretability of the discovered factors.

## 1   Introduction

There are many factors that contribute to a document's word choice: topic, syntax, sentiment, author perspective, and others. Latent variable "topic models" such as latent Dirichlet allocation (LDA) implicitly model a single factor of topical content [1]. More in-depth analyses of corpora call for models that are explicitly aware of additional factors beyond topic. Some topic models have been used to model specific factors like sentiment [2], and more general models—like the topic aspect model [3] and sparse additive generative models (SAGE) [4]—have jointly considered both topic and another factor, such as perspective. Most prior work has only considered two factors at once.[1]

This paper presents *factorial LDA*, a general framework for multi-dimensional text models that capture an arbitrary number of factors. While standard topic models associate each word token with a single latent topic variable, a multi-dimensional model associates each token with a vector of multiple factors, such as (topic, political ideology) or (product type, sentiment, author age).

Scaling to an arbitrary number of factors poses challenges that cannot be addressed with existing two-dimensional models. First, we must ensure consistency across different word distributions which have the same components. For example, the word distributions associated with the (topic, perspective) pairs (ECONOMICS,LIBERAL) and (ECONOMICS,CONSERVATIVE) should both give high probability to words about economics. Additionally, increasing the number of factors results in a multiplicative increase in the number of possible tuples that can be formed, and not all tuples will be well-supported by the data. We address these two issues by adding additional structure to our model: we impose structured word priors that link tuples with common components, and we place a sparse prior over the space of possible tuples. We demonstrate that both of these model structures lead to improvements in model performance.

In the next section, we introduce our model, where our main contributions are to:

- introduce a general model that can accommodate $K$ different factors (dimensions) of language,
- design structured priors over the word distributions that tie together common factors,
- enforce a sparsity pattern which excludes unsupported combinations of components (tuples).

We then discuss our inference procedure (§4) and share experimental results (§5).

## 2   Factorial LDA: A Multi-Dimensional Generative Model

Latent Dirichlet allocation (LDA) [1] assumes we have a set of $Z$ latent components (usually called "topics" in the context of text modeling), and each data point (a document) has a discrete distribution $\theta$ over these topics. The set of topics can be thought as a vector of length $Z$, where each cell is a pointer into a discrete distribution over words, parameterized by $\phi_z$. Under LDA, a document is generated by choosing the topic distribution $\theta$ from a Dirichlet prior, then for each token we sample a latent topic $t$ from this distribution before sampling a word $w$ from the $t$th word distribution $\phi_t$. Without additional structure, LDA tends to learn distributions which correspond to semantic topics (such as SPORTS or ECONOMICS) [6] which dominate the choice of words in a document, rather than syntax, perspective, or other aspects of document content.

Imagine that instead of a one-dimensional vector of $Z$ topics, we have a two-dimensional matrix of $Z_1$ components along one dimension (rows) and $Z_2$ components along the other (columns). This structure makes sense if a corpus is composed of two different factors, and the two dimensions might correspond to factors such as news topic and political perspective (if we are modeling newspaper editorials), or research topic and discipline (if we are modeling scientific papers). Individual cells of the matrix would represent pairs such as (ECONOMICS,CONSERVATIVE) or (GRAMMAR,LINGUISTICS) and each is associated with a word distribution $\phi_{\vec{z}}$. Conceptually, this is the idea behind the two-dimensional models of TAM [3] and SAGE [4].

Let us expand this idea further by assuming $K$ factors modeled with a $K$-dimensional array, where each cell of the array has a pointer to a word distribution corresponding to that particular $K$-tuple. For example, in addition to topic and perspective, we might want to model a third factor of the author's gender in newspaper editorials, yielding triples such as (ECONOMICS,CONSERVATIVE,MALE). Conceptually, each $K$-tuple $\vec{t}$ functions as a topic in LDA (with an associated word distribution $\phi_{\vec{t}}$) except that $K$-tuples imply a structure, e.g. the pairs (ECONOMICS,CONSERVATIVE) and (ECONOMICS,LIBERAL) are related. This is the idea behind *factorial LDA* (f-LDA).

At its core, our model follows the basic template of LDA, but each word token is associated with a $K$-tuple rather than a single topic value. Under f-LDA, each document has a distribution over tuples, and each tuple indexes into a distribution over words. Of course, without additional structure, this would simply be equivalent to LDA with $\prod_k^K Z_k$ topics. In f-LDA, we induce a factorial structure by creating priors which tie together tuples that share components: distributions involving the pair (ECONOMICS,CONSERVATIVE) should have commonalties with distributions for (ECONOMICS,LIBERAL). The key ingredients of our new model are:

- We model the intuition that tuples which share components should share other properties. For example, we expect the word distributions for (ECONOMICS,CONSERVATIVE) and (ECONOMICS,LIBERAL) to both give high probability to words about economics, while the pairs (ECONOMICS,LIBERAL) and (ENVIRONMENT,LIBERAL) should both reflect words about liberalism. Similarly, we want each document's distribution over tuples to reflect the same type of consistency. If a document is written from a liberal perspective, then we believe that pairs of the form (*,LIBERAL) are more likely to have high probability than pairs with CONSERVATIVE as the second component. This consistency across factors is encouraged by **sharing parameters** across the word and topic prior distributions in the model: this encodes our *a priori* assumption that distributions which share components should be similar.

- Additionally, we allow for **sparsity** across the set of tuples. As the dimensionality of the array increases, we are going to encounter problems of overparameterization, because the model will likely contain more tuples than are observed in the data. We handle this by having an auxiliary multi-dimensional array which encodes a sparsity pattern over tuples. The priors over tuples are augmented with this sparsity pattern. These priors model the belief that the Cartesian product of factors should be sparse; the posterior may "opt out" of some tuples.

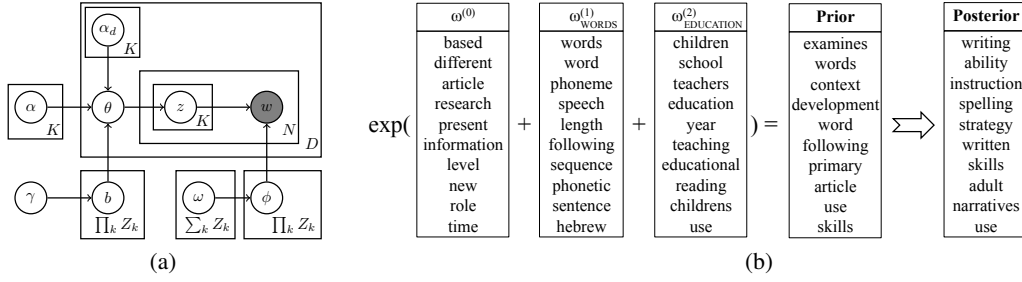

Figure 1: (a) Factorial LDA as a graphical model. (b) An illustration of word distributions in f-LDA with two factors. When applying f-LDA to a collection of scientific articles from various disciplines, we learn weights $\omega$ corresponding to a topic we call WORDS and the discipline EDUCATION as well as background words. These weights are combined to form the Dirichlet prior, and the distribution for (WORDS,EDUCATION) is drawn from this prior: this distribution describes writing education.

The generative story (we'll describe the individual pieces below) is as follows.

1. Draw the various hyperparameters $\boldsymbol{\alpha}$ and $\boldsymbol{\omega}$ from $\mathcal{N}(\mathbf{0}, \mathbf{I}\sigma^2)$

2. For each tuple $\vec{t} = (t_1, t_2, \ldots, t_K)$:

   (a) Sample word distribution $\phi_{\vec{t}} \sim \text{Dir}(\hat{\boldsymbol{\omega}}^{(\vec{t})})$
   (b) Sample sparsity "bit" $b_{\vec{t}} \sim \text{Beta}(\gamma_0, \gamma_1)$

3. For each document $d \in \mathcal{D}$:

   (a) Draw document component weights $\boldsymbol{\alpha}^{(d,k)} \sim \mathcal{N}(\mathbf{0}, \mathbf{I}\sigma^2)$ for each factor $k$
   (b) Sample distribution over tuples $\theta^{(d)} \sim \text{Dir}(\mathbf{B} \cdot \hat{\boldsymbol{\alpha}}^{(d)})$
   (c) For each token:
      i. Sample component tuple $\vec{z} \sim \theta^{(d)}$
      ii. Sample word $w \sim \phi_{\vec{z}}$

where the Dirichlet vectors $\hat{\boldsymbol{\omega}}$ and $\hat{\boldsymbol{\alpha}}$ are defined as:

$$\hat{\omega}_w^{(\vec{t})} \triangleq \exp\left\{ \omega^{(B)} + \omega_w^{(0)} + \sum_k \omega_{t_k w}^{(k)} \right\}$$

$$\hat{\alpha}_{\vec{t}}^{(d)} \triangleq \exp\left\{ \alpha^{(B)} + \left( \sum_k \alpha_{t_k}^{(\mathcal{D},k)} + \alpha_{t_k}^{(d,k)} \right) \right\} \quad (1)$$

See Figure 1a for the graphical model, and Figure 1b for an illustration of how the weight vectors $\boldsymbol{\omega}^{(0)}$ and $\boldsymbol{\omega}^{(k)}$ are combined to form $\hat{\boldsymbol{\omega}}$ for a particular tuple that was inferred by our model. The words shown have the highest weight after running our inference procedure (see §5 for experimental details).

As discussed above, the only difference between f-LDA and LDA is that structure has been added to the Dirichlet priors for the word and topic distributions. We use a form of Dirichlet-multinomial regression [7] to formulate the priors for $\phi$ and $\theta$ in terms of the log-linear functions in Eq. 1. We will now describe these priors in more detail.

**Prior over $\phi$:** We formulate the priors of $\phi$ to encourage word distributions to be consistent across components of each factor. For example, tuples that reflect the same topic should share words. To achieve this goal, we link the priors for tuples that share common components by utilizing a log-linear parameterization of the Dirichlet prior of $\phi$ (Eq. 1). Formally, we place a prior Dirichlet($\hat{\boldsymbol{\omega}}^{(\vec{t})}$) over $\phi_{\vec{t}}$, the word distribution for tuple $\vec{t} = (t_1, t_2, \ldots, t_K)$. The Dirichlet vector $\hat{\boldsymbol{\omega}}^{(\vec{t})}$ controls the precision and focus of the prior. It is a function of three types of hyperparameters. First, a single corpus-wide bias scalar $\omega^{(B)}$, and second, a vector over the vocabulary $\boldsymbol{\omega}^{(0)}$, which reflects the relative likelihood of different words. These respectively increase the overall precision of words and the default likelihood of each word. Finally, $\omega_{t_k w}^{(k)}$ introduces bias parameters for each word $w$ for component $t_k$ of the $k$th factor. By increasing the weight of a particular $\omega_{t_k w}^{(k)}$, we increase the expected relative log-probabilities of word $w$ in $\phi_{\vec{z}}$ for all $\vec{z}$ that contain component $t_k$, thereby tying these priors together.

**Prior over $\theta$:** We use a similar formulation for the prior over $\theta$. Recall that we want documents to naturally favor tuples that share components, i.e. favoring both (ECONOMICS,CONSERVATIVE) and (EDUCATION,CONSERVATIVE) if the document favors CONSERVATIVE in general. To address this, we let $\theta^{(d)}$ be drawn from Dirichlet($\hat{\boldsymbol{\alpha}}^{(d)}$), where instead of a corpus-wide prior, each document has a

vector $\hat{\boldsymbol{\alpha}}^{(d)}$ which reflects the independent contributions of the factors via a log-linear function. This function contains three types of hyperparameters. First, $\alpha^{(B)}$ is the corpus-wide precision parameter (the bias); this is shared across all documents and tuples. Second, $\alpha_{t_k}^{(\mathcal{D},k)}$ indicates the bias for the $k$th factor's component $t_k$ across the entire corpus $\mathcal{D}$, which enables the model to favor certain components *a priori*. Finally, $\alpha_{t_k}^{(d,k)}$ is the bias for the $k$th factor's component $t_k$ specifically in document $d$. This allows documents to favor certain components over others, such as the perspective CONSERVATIVE in a specific document. We assume all $\omega$s and $\alpha$s are independent and normally distributed around 0, which gives us L2 regularization during optimization.

**Sparsity over tuples:** Finally, we describe the generation of the sparsity pattern over tuples in the corpus. We assume a $K$-dimensional binary array $\mathbf{B}$, where an entry $b_{\vec{t}}$ corresponds to tuple $\vec{t}$. If $b_{\vec{t}} = 1$, then $\vec{t}$ is active: that is, we are allowed to chose $\vec{t}$ to generate a token and we learn $\phi_{\vec{t}}$; otherwise we do not. We modify this prior over $\theta$ to include a binary mask of the tuples: $\theta^{(d)} \sim \text{Dirichlet}(\mathbf{B} \cdot \hat{\boldsymbol{\alpha}}^{(d)})$, where $\cdot$ is the Hadamard (cell-wise) product. $\theta$ will not include tuples for which $b_{\vec{t}} = 0$; otherwise the prior will remain unchanged.

We would ideally model $\mathbf{B}$ so that its values are in $\{0,1\}$. While we could use a Beta-Bernoulli model (a finite Indian Buffet Process [8]) to generate a finite binary matrix (array), this model is typically learned over continuous data; learning over discrete observations (tuples) can be exceedingly difficult since forcing the model to change a bit can yield large changes to the observations, which makes mixing very slow.[2] To aid learning, we relax the constraint that $\mathbf{B}$ must be binary and instead allow $b_{\vec{t}}$ to be real-valued in $(0,1)$. This is a common approximation used in other models, such as artificial neural networks and deep belief networks. To encourage sparsity, we place a "U-shaped" $\text{Beta}(\gamma_0, \gamma_1)$ prior over $b_{\vec{t}}$, where $\gamma < 1$, which yields a density function that is concentrated around the edges 0 and 1. Empirically, we will show that this effectively learns a sparse binary $\mathbf{B}$. The effect is that the prior assigns tiny probabilities to some tuples instead of strictly 0.

## 3  Related Work

Previous work on multi-dimensional modeling includes the topic aspect model (TAM) [3], multi-view LDA (mv-LDA) [10], cross-collection LDA [11] and sparse additive generative models (SAGE) [4], which jointly consider both topic and another factor. Other work has jointly modeled topic and sentiment [2]. Zhang et al. [12] apply PLSA [13] to multi-dimensional OLAP data, but not with a joint model. Our work is the first to jointly model an arbitrary number of factors. A rather different approach considered different dimensions of clustering using spectral methods [14], in which $K$ different clusterings are obtained by considering $K$ different eigenvectors. For example, product reviews can be clustered not only by topic, but also by sentiment and author attributes.

We contrast this body of work with probabilistic matrix and tensor factorization models [15, 16] which model data that has already been organized in multiple dimensions – for example, topic-like models have been used to model the movie ratings within a matrix of users and movies. f-LDA and the models described above, however, operate over flat input (text documents), and it is only the latent structure that is assumed to be organized along multiple dimensions.

An important contribution of f-LDA is the use of priors to tie together word distributions with the same components. Previous work with two-dimensional models, such as TAM and mv-LDA, assume conditional independence among all $\phi$, and there is no explicit encouragement of correlation. An alternative approach would be to strictly enforce consistency, such as through a "product of experts" model [17], in which each factor has independent word distributions that are multiplied together and renormalized to form the distribution for a particular tuple, i.e. $\phi_{\vec{t}} \propto \prod_k \phi_{t_k}$. Syntactic topic models [18] and shared components topic models [19] follow this approach. Our structured word prior generalizes both of these approaches. By setting all $\boldsymbol{\omega}^{(k)}$ to 0, the factors have no influence on the prior and we obtain the distribution independence of TAM. If instead we have large $\omega$ values, then the model behaves like a product of experts; as precision increases, the posterior converges to the prior. By **learning** $\omega$ our model can determine the optimal amount of coherence among the $\phi$.

Another key part of f-LDA is the inclusion of a sparsity pattern. There have been several recent approaches that enforce sparsity in topic models. Various applications of sparsity can be organized into three categories. First, one could enforce sparsity over the topic-specific word distributions, forcing each topic to select a subset of relevant words. This is the idea behind sparse topic models [20], which restrict topics to a subset of the vocabulary, and SAGE [4], which applies L1 regularization to word weights. A second approach is to enforce sparsity in the document-specific topic distributions, focusing each document on a subset of relevant topics. This is the idea in focused topic models [9]. Finally—our contribution—is to impose sparsity among the set of topics (or $K$-tuples) that are available to the model. Among sparsity-inducing regularizers, one that closely relates to our goals is the *group lasso* [21]. While the standard lasso will drive vector elements to 0, the group lasso will drive entire vectors to 0.

## 4 Inference and Optimization

f-LDA turns out to be fairly similar to LDA in terms of inference. In both models, words are generated by first sampling a latent variable (in our case, a latent tuple) from a distribution $\theta$, then sampling the word from $\phi$ conditioned on the latent variable. The differences between LDA and f-LDA lie in the parameters of the Dirichlet priors. The presentation of our optimization procedure focuses on these parameters.

We follow the common approach of alternating between sampling the latent variables and direct optimization of the Bayesian hyperparameters [22]. We use a Gibbs sampler to estimate $\mathbb{E}[\vec{\mathbf{z}}]$, and given the current estimate of this expectation, we optimize the parameters $\boldsymbol{\alpha}$, $\boldsymbol{\omega}$ and $\mathbf{B}$. These two steps form a Monte Carlo EM (MCEM) routine.

### 4.1 Latent Variable Sampling

The latent variables $\vec{z}$ are sampled using the standard collapsed Gibbs sampler for LDA [23], with the exception that the basic Dirichlet priors have been replaced with our structured priors for $\theta$ and $\phi$. The sampling equation for $\vec{z}$ for token $i$, given all other latent variable assignments $\vec{\mathbf{z}}$, the corpus $\mathbf{w}$ and the parameters ($\boldsymbol{\alpha}$, $\boldsymbol{\omega}$, and $\mathbf{B}$) becomes:

$$p(\vec{z_i} = \vec{t} \,|\, \vec{\mathbf{z}} \setminus \{\vec{z_i}\}, \mathbf{w}, \boldsymbol{\alpha}, \boldsymbol{\omega}, \mathbf{B}) \propto \left(n_{\vec{t}}^d + b_{\vec{t}}\,\hat{\alpha}_{\vec{t}}^{(d)}\right)\left(\frac{n_w^{\vec{t}} + \hat{\omega}_w^{(\vec{t})}}{\sum_{w'} n_{w'}^{\vec{t}} + \hat{\omega}_{w'}^{(\vec{t})}}\right) \tag{2}$$

where $n_a^b$ denotes the number of times $a$ occurs in $b$.

### 4.2 Optimizing the Sparsity Array and Hyperparameters

For mathematical convenience, we reparameterize $\mathbf{B}$ in terms of the logistic function $\sigma$, such that $b_{\vec{t}} \equiv \sigma(\beta_{\vec{t}})$. We optimize $\beta \in \mathbb{R}$ to obtain $b \in (0, 1)$. The derivative of $\sigma(x)$ has the simple form $\sigma(x)\sigma(-x)$. For a tuple $\vec{t}$, the gradient of the corpus log likelihood $\mathcal{L}$ with respect to $\beta_{\vec{t}}$ is:

$$\frac{\partial \mathcal{L}}{\partial \beta_{\vec{t}}} = (\gamma_0 - 1)\sigma(-\beta_{\vec{t}}) + (\gamma_1 - 1)(-\sigma(\beta_{\vec{t}})) + \left[\sum_{d \in \mathcal{D}} \sigma(\beta_{\vec{t}})\sigma(-\beta_{\vec{t}})\,\hat{\alpha}_{\vec{t}}^{(d)} \times \right. \tag{3}$$

$$\left. \left(\Psi(n_{\vec{t}}^d + \sigma(\beta_{\vec{t}})\hat{\alpha}_{\vec{t}}^{(d)}) - \Psi(\sigma(\beta_{\vec{t}})\hat{\alpha}_{\vec{t}}^{(d)}) + \Psi\left(\sum_{\vec{u}} \sigma(\beta_{\vec{u}})\,\hat{\alpha}_{\vec{u}}^{(d)}\right) - \Psi\left(\sum_{\vec{u}} n_{\vec{u}}^d + \sigma(\beta_{\vec{u}})\,\hat{\alpha}_{\vec{u}}^{(d)}\right)\right)\right]$$

where the $\gamma$ values are the Beta parameters. The top terms are a result of the Beta prior over $b_{\vec{t}}$, while the summation over documents reflects the gradient of the Dirichlet-multinomial compound. Standard non-convex optimization methods can be used on this gradient. To avoid shallow local minima, we optimize this gradually by taking small gradient steps, performing a single iteration of gradient ascent after each Gibbs sampling iteration (see §5 for more details).

The gradients for the $\boldsymbol{\alpha}$ and $\boldsymbol{\omega}$ variables have a similar form to (3); the main difference with $\boldsymbol{\omega}$ is that the gradient involves a sum over components rather than over documents. We similarly update these values through gradient ascent.

# 5 Experiments

We experiment with two data sets that could contain multiple factors. The first is a collection of 5000 computational linguistics abstracts from the ACL Anthology (ACL). The second combines these abstracts (C) with several journals in the fields of linguistics (L), education (E), and psychology (P). We use 1000 articles from each discipline (CLEP). For both corpora, we keep an additional 1000 documents for development and 1000 for test (uniformly representative of the 4 CLEP disciplines).

We used $\vec{Z} = (*, 2, 2)$ for ACL and $\vec{Z} = (*, 4)$ for CLEP for various numbers of "topics" $Z_1 \in \{5, \ldots, 50\}$. While we cannot say in advance what each factor will represent, we observed that when $Z_k$ is large, components along this factor correspond to topics. Therefore, we set $Z_1 > Z_{k>1}$ and assume the first factor is topic. While our model presentation assumed latent factors, we could observe factors, such as knowing the journal of each article in CLEP. However, our experiments strictly focus on the **unsupervised** setting to measure what the model can infer on its own.

We will compare our complete model against simpler models by ablating parts of f-LDA. If we remove the structured word priors and array sparsity, we are left with a basic multi-dimensional model (base). We will compare against models where we add back in the structured word priors (W) and array sparsity (S), and finally the full f-LDA model (SW). All variants are identical except that we fix all $\boldsymbol{\omega}^{(k)} = \mathbf{0}$ to remove structured word priors and fix $\mathbf{B} = \mathbf{1}$ to remove sparsity.

We also compare against the topic aspect model (TAM) [3], a two-dimensional model, using the public implementation.[3] TAM is similar to the "base" two-factor f-LDA model except that f-LDA has a single $\theta$ per document with priors that are independently weighted by each factor, whereas TAM has $K$ independent $\theta$s, with a different $\theta_k$ for each factor. If the Dirichlet precision in f-LDA is very high, then it should exhibit similar behavior as having separate $\theta$s. TAM only models two dimensions so we are restricted to running it on the two-dimensional CLEP data set.

For hyperparameters, we set $\gamma_0 = \gamma_1 = 0.1$ in the Beta prior over $b_{\vec{t}}$, and we set $\sigma^2 = 10$ for $\alpha$ and 1 for $\omega$ in the Gaussian prior over weights. Bias parameters ($\alpha^{(B)}$, $\omega^{(B)}$) are initialized to $-5$ for weak initial priors. Our sampling algorithm alternates between a full pass over tokens and a single gradient step on the parameters (step size of $10^{-2}$ for $\alpha$; $10^{-3}$ for $\omega$ and $\beta$). Results are averaged or pooled from five trials of randomly initialized chains, which are each run for 10,000 iterations.

**Perplexity**    Following standard practice we measure perplexity on held-out data by fixing all parameters during training except document-specific parameters ($\alpha^{(d,k)}$, $\theta^{(d)}$), which are computed from the test document. We use the "document completion" method: we infer parameters from half a document and measure perplexity on the remaining half [24]. Monte Carlo EM is run on test data for 200 iterations. Average perplexity comes from another 10 iterations.

Figure 2a shows that the structured word priors yield lower perplexity, while results for sparse models are mixed. On ACL, sparsity consistently improves perplexity once the number of topics exceeds 20, while on CLEP sparsity does worse. Experiments with varying $K$ yielded similar orderings, suggesting that differences are data dependent and not dependent on $K$. On CLEP, we find that TAM performs worse than f-LDA with a lower number of topics (which is what we find to work best qualitatively), but catches up as the number of topics increases. (Beyond 50 topics, we find that TAM's perplexity stays about the same, and then begins to increase again once $Z \geq 75$.) Thus, in addition to scaling to more factors, f-LDA is more predictive than simpler multi-dimensional models.

**Qualitative Results**    To illustrate model behavior we include a sample of output on ACL (Figure 3). We consider the component-specific weights for each factor $\vec{\omega}_{t_k}^{(k)}$, which present an "overview" of each component, as well as the tuple-specific word distributions $\phi_{\vec{t}}$. Upon examination, we determined that the first factor ($Z_1 = 20$) corresponds to topic, the second ($Z_2 = 2$) to approach (empirical vs. theoretical), and the third ($Z_3 = 2$) to focus (methods vs. applications). The top row shows words common across all components for each factor. The bottom row shows specific $\phi_{\vec{t}}$. Consider the topic SPEECH: the triple (SPEECH,METHODS,THEORETICAL) emphasizes the linguistic side of speech processing (*phonological*, *prosodic*, etc.) while (SPEECH,APPLICATIONS,EMPIRICAL) is predominantly about dialogue systems and speech interfaces. We also see tuple sparsity (shaded

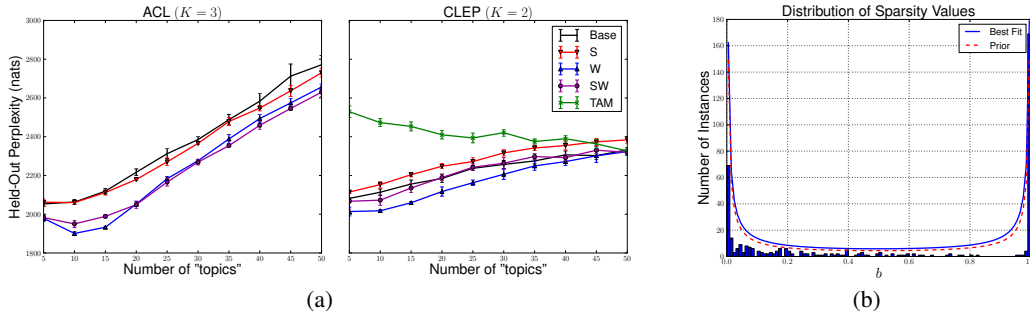

(a)                                                           (b)

Figure 2: (a) The document completion perplexity on two data sets. Models with "W" use structured word priors, and those with "S" use sparsity. Error bars indicate 90% confidence intervals. When pooling results across all numbers of topics $\geq 20$, we find that S is significantly better than Base with $p = 1.4 \times 10^{-4}$ and SW is better than W with $p = 5 \times 10^{-5}$ on the ACL corpus. (b) The distribution of sparsity values induced on the ACL corpus with $\vec{Z} = (20, 2, 2)$.

|  | ACL | CLEP | ACL | CLEP |
|---|---|---|---|---|
|  | Intrusion Accuracy | | Relatedness Score (1–5) | |
| TAM | n/a | 46% | n/a | $2.29 \pm 0.26$ |
| Baseline | 39% | 38% | $2.35 \pm 0.31$ | $2.55 \pm 0.37$ |
| Sparsity (S) | 51% | 43% | $2.61 \pm 0.37$ | $2.53 \pm 0.48$ |
| Word Priors (W) | **76%** | 45% | $3.56 \pm 0.36$ | $2.59 \pm 0.33$ |
| Combined (SW) | 73% | **67%** | **3.90** $\pm 0.37$ | **2.67** $\pm 0.55$ |

Table 1: Results from human judgments. The best scoring model for each data set is in bold. 90% confidence intervals are indicated for scores; scores were more varied on the CLEP corpus.

tuples, in which $b_{\vec{t}} \leq 0.5$) for poor tuples. For example, under the topic of DATA, a mostly empirical topic, tuples along the THEORETICAL component are inactive.

**Human Judgments**   Perplexity may not correlate with human judgments [6], which are important for f-LDA since structured word priors and array sparsity are motivated in part by semantic coherence. We measured interpretability based on the notion of relatedness: among components that are inferred to belong to the same factor, how many actually make sense together? Seven annotators provided judgments for two related tasks. First, we presented annotators with two word lists (ten most frequent words assigned to each tuple[4]) that are assigned to the same topic, along with a word list randomly selected from another topic. Annotators are asked to choose the word list that does not belong, i.e. an intrusion test [6]. If the two tuples from the same topic are strongly related, the random list should be easy to identify. Second, annotators are presented with pairs of word lists from the same topic and asked to judge the degree of relation using a 5-point Likert scale.

We ran these experiments on both corpora with 20 topics. For the two models without the structured word priors, we use a symmetric prior (by optimizing only $\omega^{(B)}$ and fixing $\omega^{(0)} = \mathbf{0}$), since symmetric word priors can lead to better interpretability [22].[5] We exclude tuples with $b_{\vec{t}} \leq 0.5$.

Across all data sets and models, annotators labeled 362 triples in the intrusion experiment and 333 pairs in the scoring experiment. The results (Table 1) differ slightly from the perplexity results. The word priors help in all cases, but much more so on ACL. The models with sparsity are generally better than those without, even on CLEP, in contrast to perplexity where sparse models did worse. This suggests that removing tuples with small $b_{\vec{t}}$ values removes nonsensical tuples. Overall, the judgments are worse for the CLEP corpus; this appears to be a difficult corpus to model due to high topic diversity and low overlap across disciplines. TAM is judged to be worse than all f-LDA variants when directly scored by annotators. The intrusion performance with TAM is better than or comparable to the ablated versions of f-LDA, but worse than the full model. It thus appears that both the structured priors and sparsity yield more interpretable word clusters.

| "Topic" | | | | "Approach" | | "Focus" | |
|---|---|---|---|---|---|---|---|
| "SPEECH" | "I.R." | "M.T." | ... | "EMPIRICAL" | "THEORETICAL" | "METHODS" | "APPLICATIONS" |
| speech | document | translation | ... | task | theory | word | user |
| spoken | retrieval | machine | ... | tasks | description | algorithm | research |
| recognition | documents | source | ... | performance | formal | method | project |
| state | question | mt | ... | improve | forms | accuracy | technology |
| vocabulary | web | parallel | ... | accuracy | treatment | best | processing |
| recognizer | answering | french | ... | learning | linguistics | sentence | science |
| utterances | query | bilingual | ... | demonstrate | syntax | statistical | natural |
| synthesis | answer | transfer | ... | using | ed | previously | development |

| Topic | SPEECH | | DATA | | MODELING | | GRAMMAR | |
|---|---|---|---|---|---|---|---|---|
| Focus | METHODS | APPL. | METHODS | APPL. | METHODS | APPL. | METHODS | APPL. |
| EMPIRICAL | (b=0.20) | (b=1.00) dialogue spoken speech dialogues understanding task recognition | (b=1.00) corpus data training model tagging annotated test | (b=1.00) data corpus annotation annotated corpora collection xml | (b=1.00) models model approach shown error errors statistical | (b=0.50) | (b=1.00) parsing parser syntactic tree parse dependency treebank | (b=0.57) grammar parsing based robust component processing linguistic |
| THEORETICAL | (b=0.99) speech words recognition prosodic written phonological spoken | (b=0.00) | (b=0.07) | (b=0.02) | (b=1.00) rules rule model shown models right left | (b=0.01) | (b=1.00) grammar parsing grammars structures paper formalism based | (b=1.00) grammar grammars formalism parsing based efficient unification |

Figure 3: Example output from the ACL corpus with $\vec{Z} = (20, 2, 2)$. Above: The top words (based on their $\omega$ values) for a few components from three factors. Below: A three-dimensional table showing a sample of four topics (i.e. components of the first factor) with their top words (based on their $\phi$ values) as they appear in all combinations of factors. The components in the top table are combined to create 3-tuples in the bottom table. Shaded cells ($b \leq 0.5$) are inactive. The names of factors and their components in quotes are manually assigned through post-hoc analysis.

**Sparsity Patterns**   Finally, we examine the learned sparsity patterns: how much of **B** is close to 0 or 1? Figure 2b shows a histogram of $b_{\vec{t}}$ values (ACL with 20 topics, 3 factors) pooled across five sampling chains. The majority of values are close to 0 or 1, effectively capturing a sparse binary array. The higher variance near 0 relative to 1 suggests that the model prefers to keep bits "on"— and give tuples tiny probability—rather than "off." This suggests that a model with a hard constraint might struggle to "turn off" bits during inference.

While we fixed the Beta parameters in our experiments, these can be tuned to control sparsity. The model will favor more "on" than "off" bits by setting $\gamma_1 > \gamma_0$, or vice versa. When $\gamma > 1$, the Beta distribution no longer favors sparsity; we confirmed empirically that this leads to $b_{\vec{t}}$ values that are closer to 0.8 or 0.9 rather than 1. In contrast, setting $\gamma \ll 0.1$ yields more extreme values near 0 and 1 than with $\gamma = 0.1$ (e.g. .9999 instead of .991), but this does not greatly affect the number of non-binary values. Thus, a sparse prior alone cannot fully satisfy our preference that **B** is binary.

**Comparison to LDA**   The runtimes of samplers for LDA and f-LDA are on the same order (but we have not investigated differences in mixing time). Our f-LDA implementation is one to two times slower per iteration than our own comparable LDA implementation (with hyperparameter optimization using the methods in [25]). We did not observe a consistent pattern regarding the perplexity of the two models. Averaged across all numbers of topics, the perplexity of LDA was 97% the perplexity of f-LDA on ACL and 104% on CLEP. Note that our experiments always use a comparable number of word distributions, thus $\vec{Z} = (20, 2, 2)$ is the same as $Z = 80$ topics in LDA.

# 6   Conclusion

We have presented factorial LDA, a multi-dimensional text model that can incorporate an arbitrary number of factors. To encourage the model to learn the desired patterns, we developed two new types of priors: word priors that share features across factors, and a sparsity prior that restricts the set of active tuples. We have shown both qualitatively and quantitatively that f-LDA is capable of discovering interpretable patterns even in multi-dimensional spaces.

## Acknowledgements

We are grateful to Jason Eisner, Matthew Gormley, Nicholas Andrews, David Mimno, and the anonymous reviewers for helpful discussions and feedback. This work was supported in part by a National Science Foundation Graduate Research Fellowship under Grant No. DGE-0707427.

## Footnotes

[1] A recent variant of SAGE modeled three factors in historic documents: topic, time, and location [5].

[2] One approach is to (approximately) collapse out the sparsity array [9], but this is difficult when working over the entire corpus of tokens. Experiments with Metropolis-Hastings samplers, split-merge based samplers, and alternative prior structures all suffered from mixing problems.

[3]Most other two-dimensional models, including SAGE [4] and multi-view LDA [10], assume that the second factor is fixed and observed. Our focus in this paper is fully unsupervised models.

[4]We use frequency instead of the actual posterior because including the learned priors (which share many words) could make the task unfairly easy.

[5]We used an asymmetric prior for the perplexity experiments, which gave slightly better results.

## References

[1] D. Blei, A. Ng, and M. Jordan. Latent Dirichlet allocation. *JMLR*, 2003.

[2] Q. Mei, X. Ling, M. Wondra, H. Su, and C. Zhai. Topic sentiment mixture: modeling facets and opinions in weblogs. In *WWW*, 2007.

[3] M. Paul and R. Girju. A two-dimensional topic-aspect model for discovering multi-faceted topics. In *AAAI*, 2010.

[4] J. Eisenstein, A. Ahmed, and E. P. Xing. Sparse additive generative models of text. In *ICML*, 2011.

[5] W. Y. Wang, E. Mayfield, S. Naidu, and J. Dittmar. Historical analysis of legal opinions with a sparse mixed-effects latent variable model. In *ACL*, pages 740–749, July 2012.

[6] J. Chang, J. Boyd-Graber, S. Gerrish, C. Wang, and D. Blei. Reading tea leaves: How humans interpret topic models. In *NIPS*, 2009.

[7] D. Mimno and A. McCallum. Topic models conditioned on arbitrary features with dirichlet-multinomial regression. In *UAI*, 2008.

[8] T. Griffiths and Z. Ghahramani. Infinite latent feature models and the Indian buffet process. In *NIPS*, 2006.

[9] S. Williamson, C. Wang, K. Heller, and D. Blei. The IBP-compound dirichlet process and its application to focused topic modeling. In *ICML*, 2010.

[10] A. Ahmed and E. P. Xing. Staying informed: supervised and semi-supervised multi-view topical analysis of ideological perspective. In *EMNLP*, pages 1140–1150, 2010.

[11] M. Paul and R. Girju. Cross-cultural analysis of blogs and forums with mixed-collection topic models. In *EMNLP*, pages 1408–1417, August 2009.

[12] D. Zhang, C. Zhai, J. Han, A. Srivastava, and N. Oza. Topic modeling for OLAP on multidimensional text databases: topic cube and its applications. *Statistical Analysis and Data Mining*, 2, 2009.

[13] T. Hofmann. Probabilistic latent semantic indexing. In *SIGIR*, 1999.

[14] S. Dasgupta and V. Ng. Mining clustering dimensions. In *ICML*, 2010.

[15] I. Porteous, E. Bart, and M. Welling. Multi-HDP: a non parametric Bayesian model for tensor factorization. In *AAAI*, pages 1487–1490, 2008.

[16] L. Mackey, D. Weiss, and M. I. Jordan. Mixed membership matrix factorization. In *ICML*, 2010.

[17] G. E. Hinton. Training products of experts by minimizing contrastive divergence. *Neural Comput.*, 14:1771–1800, August 2002.

[18] J. Boyd-Graber and D. Blei. Syntactic topic models. In *NIPS*, 2008.

[19] M. R. Gormley, M. Dredze, B. Van Durme, and J. Eisner. Shared components topic models. In *NAACL*, 2010.

[20] C. Wang and D. Blei. Decoupling sparsity and smoothness in the discrete hierarchical Dirichlet process. In *NIPS*, 2009.

[21] L. Meier, S. van de Geer, and P. Bühlmann. The group lasso for logistic regression. *Journal Of The Royal Statistical Society Series B*, 70(1):53–71, 2008.

[22] H. Wallach, D. Mimno, and A. McCallum. Rethinking LDA: Why priors matter. In *NIPS*, 2009.

[23] T. Griffiths and M. Steyvers. Finding scientific topics. In *Proceedings of the National Academy of Sciences of the United States of America*, 2004.

[24] M. Rosen-Zvi, T. Griffiths, M. Steyvers, and P. Smyth. The author-topic model for authors and documents. In *UAI*, 2004.

[25] Michael J. Paul. Mixed membership Markov models for unsupervised conversation modeling. In *EMNLP-CoNLL*, 2012.

